# S-Map: A network with a simple self-organization algorithm for generative topographic mappings

**Kimmo Kiviluoto**
Laboratory of Computer and
Information Science
Helsinki University of Technology
P.O. Box 2200
FIN-02015 HUT, Espoo, Finland
Kimmo.Kiviluoto@hut.fi

**Erkki Oja**
Laboratory of Computer and
Information Science
Helsinki University of Technology
P.O. Box 2200
FIN-02015 HUT, Espoo, Finland
Erkki.Oja@hut.fi

## Abstract

The S-Map is a network with a simple learning algorithm that combines the self-organization capability of the Self-Organizing Map (SOM) and the probabilistic interpretability of the Generative Topographic Mapping (GTM). The simulations suggest that the S-Map algorithm has a stronger tendency to self-organize from random initial configuration than the GTM. The S-Map algorithm can be further simplified to employ pure Hebbian learning, without changing the qualitative behaviour of the network.

## 1   Introduction

The self-organizing map (SOM; for a review, see [1]) forms a topographic mapping from the data space onto a (usually two-dimensional) output space. The SOM has been succesfully used in a large number of applications [2]; nevertheless, there are some open theoretical questions, as discussed in [1, 3]. Most of these questions arise because of the following two facts: the SOM is not a generative model, i.e. it does not generate a density in the data space, and it does not have a well-defined objective function that the training process would strictly minimize.

Bishop et al. [3] introduced the generative topographic mapping (GTM) as a solution to these problems. However, it seems that the GTM requires a careful initialization to self-organize. Although this can be done in many practical applications, from a theoretical point of view the GTM does not yet offer a fully satisfactory model for natural or artificial self-organizing systems.

In this paper, we first briefly review the SOM and GTM algorithms (section 2); then we introduce the S-Map, which may be regarded as a crossbreed of SOM and GTM (section 3); finally, we present some simulation results with the three algorithms (section 4), showing that the S-Map manages to combine the computational simplicity and the ability to self-organize of the SOM with the probabilistic framework of the GTM.

## 2   SOM and GTM

### 2.1   The SOM algorithm

The self-organizing map associates each data vector $\boldsymbol{\xi}^t$ with that map unit that has its weight vector closest to the data vector. The *activations* $\eta_i^t$ of the map units are given by

$$\eta_i^t = \begin{cases} 1, & \text{when } \|\boldsymbol{\mu}_i - \boldsymbol{\xi}^t\| < \|\boldsymbol{\mu}_j - \boldsymbol{\xi}^t\|, \ \forall j \neq i \\ 0, & \text{otherwise} \end{cases} \tag{1}$$

where $\boldsymbol{\mu}_i$ is the weight vector of the $i^{\text{th}}$ map unit $\boldsymbol{\zeta}_i$, $i = 1, \ldots, K$. Using these activations, the SOM weight vector update rule can be written as

$$\boldsymbol{\mu}_j := \boldsymbol{\mu}_j + \delta^t \sum_{i=1}^{K} \eta_i^t h(\boldsymbol{\zeta}_i, \boldsymbol{\zeta}_j; \beta^t)(\boldsymbol{\xi}^t - \boldsymbol{\mu}_j) \tag{2}$$

Here parameter $\delta^t$ is a learning rate parameter that decreases with time. The *neighborhood function* $h(\boldsymbol{\zeta}_i, \boldsymbol{\zeta}_j; \beta^t)$ is a decreasing function of the distance between map units $\boldsymbol{\zeta}_i$ and $\boldsymbol{\zeta}_j$; $\beta^t$ is a width parameter that makes the neighborhood function get narrower as learning proceeds. One popular choice for the neighborhood function is a Gaussian with inverse variance $\beta^t$.

### 2.2   The GTM algorithm

In the GTM algorithm, the map is considered as a *latent space*, from which a nonlinear mapping to the data space is first defined. Specifically, a point $\boldsymbol{\zeta}$ in the latent space is mapped to the point $\boldsymbol{v}$ in the data space according to the formula

$$\boldsymbol{v}(\boldsymbol{\zeta}; \mathbf{M}) = \mathbf{M}\boldsymbol{\phi}(\boldsymbol{\zeta}) = \sum_{j=1}^{L} \phi_j(\boldsymbol{\zeta})\boldsymbol{\mu}_j \tag{3}$$

where $\boldsymbol{\phi}$ is a vector consisting of $L$ Gaussian basis functions, and $\mathbf{M}$ is a $D \times L$ matrix that has vectors $\boldsymbol{\mu}_j$ as its columns, $D$ being the dimension of the data space.

The probability density $p(\boldsymbol{\zeta})$ in the latent space generates a density to the manifold that lies in the data space and is defined by (3). If the latent space is of lower dimension than the data space, the manifold would be singular, so a Gaussian noise model is added. A single point in the latent space generates thus the following density in the data space:

$$p(\boldsymbol{\xi}|\boldsymbol{\zeta}; \mathbf{M}, \beta) = \left(\frac{\beta}{2\pi}\right)^{D/2} \exp\left[-\frac{\beta}{2}\|\boldsymbol{v}(\boldsymbol{\zeta}; \mathbf{M}) - \boldsymbol{\xi}\|^2\right] \tag{4}$$

where $\beta$ is the inverse of the variance of the noise.

The key point of the GTM is to approximate the density in the data space by assuming the latent space prior $p(\boldsymbol{\zeta})$ to consist of equiprobable delta functions that

form a regular lattice in the latent space. The centers $\zeta_i$ of the delta functions are called the *latent vectors* of the GTM, and they are the GTM equivalent to the SOM map units. The approximation of the density generated in the data space is thus given by

$$p(\xi|\mathbf{M},\beta) = \frac{1}{K}\sum_{i=1}^{K} p(\xi|\zeta_i;\mathbf{M},\beta) \tag{5}$$

The parameters of the GTM are determined by minimizing the negative log likelihood error

$$\mathcal{E}(\mathbf{M},\beta) = -\sum_{t=1}^{T}\ln\left[\frac{1}{K}\sum_{i=1}^{K} p(\xi^t|\zeta_i;\mathbf{M},\beta)\right] \tag{6}$$

over the set of sample vectors $\{\xi^t\}$. The batch version of the GTM uses the EM algorithm [4]; for details, see [3]. One may also resort to an on-line gradient descent procedure that yields the GTM update steps

$$\mu_j^{t+1} := \mu_j^t + \delta^t\beta^t\sum_{i=1}^{K}\eta_i^t(\mathbf{M}^t,\beta^t)\phi_j(\zeta_i)[\xi^t - v(\zeta_i;\mathbf{M}^t)] \tag{7}$$

$$\beta^{t+1} := \beta^t + \delta^t\left[\frac{1}{2}\sum_{i=1}^{K}\eta_i^t(\mathbf{M}^t,\beta^t)\|\xi^t - v(\zeta_i;\mathbf{M}^t)\|^2 - \frac{D}{2\beta^t}\right] \tag{8}$$

where $\eta_i^t(\mathbf{M},\beta)$ is the GTM counterpart to the SOM unit activation, the posterior probability $p(\zeta_i|\xi^t;\mathbf{M},\beta)$ of the latent vector $\zeta_i$ given data vector $\xi^t$:

$$\begin{aligned}
\eta_i^t(\mathbf{M},\beta) &= p(\zeta_i|\xi^t;\mathbf{M},\beta) \\
&= \frac{p(\xi^t|\zeta_i;\mathbf{M},\beta)}{\sum_{i'=1}^{K} p(\xi^t|\zeta_{i'};\mathbf{M},\beta)} \\
&= \frac{\exp[-\frac{\beta}{2}\|v(\zeta_i;\mathbf{M}) - \xi^t\|^2]}{\sum_{i'=1}^{K}\exp[-\frac{\beta}{2}\|v(\zeta_{i'};\mathbf{M}) - \xi^t\|^2]}
\end{aligned} \tag{9}$$

### 2.3 Connections between SOM and GTM

Let us consider a GTM that has an equal number of latent vectors and basis functions[1], each latent vector $\zeta_i$ being the center for one Gaussian basis function $\phi_i(\zeta)$. Latent vector locations may be viewed as units of the SOM, and consequently the basis functions may be interpreted as connection strengths between the units. Let us use the shorthand notation $\phi_j^i \equiv \phi_j(\zeta_i)$. Note that $\phi_j^i = \phi_i^j$, and assume that the basis functions be normalized so that $\sum_{j=1}^{K}\phi_j^i = \sum_{i=1}^{K}\phi_j^i = 1$.

At the zero-noise limit, or when $\beta \to \infty$, the softmax activations of the GTM given in (9) approach the winner-take-all function (1) of the SOM. The winner unit $\zeta_{c(t)}$ for the data vector $\xi^t$ is the map unit that has its *image* closest to the data vector, so that the index $c(t)$ is given by

$$c(t) = \operatorname*{argmin}_i \|v(\zeta_i) - \xi^t\| = \operatorname*{argmin}_i \left\|\left(\sum_{j=1}^{K}\phi_j^i\mu_j\right) - \xi^t\right\| \tag{10}$$

The GTM weight update step (7) then becomes

$$\mu_j^{t+1} := \mu_j^t + \delta^t \phi_j^{c(t)} [\xi^t - v(\zeta_{c(t)}; M^t)] \tag{11}$$

This resembles the variant of SOM, in which the winner is searched with the rule (10) and weights are updated as

$$\mu_j^{t+1} := \mu_j^t + \delta^t \phi_j^{c(t)} (\xi^t - \mu_j^t) \tag{12}$$

Unlike the original SOM rules (1) and (2), the modified SOM with rules (10) and (12) does minimize a well-defined objective function: the *SOM distortion measure* [5, 6, 7, 1]. However, there is a difference between GTM and SOM learning rules (11) and (12). With SOM, each individual weight vector moves towards the data vector, but with GTM, the *image* of the winner latent vector $v(\zeta_{c(t)}; M)$ moves towards the data vector, and all weight vectors $\mu_j$ move to the same direction.

For nonzero noise, when $0 < \beta < \infty$, there is more difference between GTM and SOM: with GTM, not only the winner unit but activations from other units as well contribute to the weight update.

## 3   S-Map

Combining the softmax activations of the GTM and the learning rule of the SOM, we arrive at a new algorithm: the S-Map.

### 3.1   The S-Map algorithm

The S-Map resembles a GTM with an equal number of latent vectors and basis functions. The position of the $i^{\text{th}}$ unit on the map is is given by the latent vector $\zeta_i$; the connection strength of the unit to another unit $j$ is $\phi_j^i$, and a weight vector $\mu_i$ is associated with the unit. The activation of the unit is obtained using rule (9).

The S-Map weights learn proportionally to the activation of the unit that the weight is associated with, and the activations of the neighboring units:

$$\mu_j^{t+1} := \mu_j^t + \delta^t \left( \sum_{i=1}^K \phi_j^i \eta_i^t \right) (\xi^t - \mu_j^t) \tag{13}$$

which can be further simplified to a fully Hebbian rule, updating each weight proportionally to the activation of the corresponding unit only, so that

$$\mu_j^{t+1} := \mu_j^t + \delta^t \eta_j^t (\xi^t - \mu_j^t) \tag{14}$$

The parameter $\beta$ value may be adjusted in the following way: start with a small value, slowly increase it so that the map unfolds and spreads out, and then keep increasing the value as long as the error (6) decreases. The parameter adjustment scheme could also be connected with the topographic error of the mapping, as proposed in [9] for the SOM.

Assuming normalized input and weight vectors, the "dot-product metric" form of the learning rules (13) and (14) may be written as

$$\mu_j^{t+1} := \mu_j^t + \delta^t \left( \sum_{i=1}^K \phi_j^i \eta_i^t \right) (I - \mu_j^t \mu_j^{tT}) \xi^t \tag{15}$$

and

$$\boldsymbol{\mu}_j^{t+1} := \boldsymbol{\mu}_j^t + \delta^t \eta_j^t (\mathbf{I} - \boldsymbol{\mu}_j^t \boldsymbol{\mu}_j^{tT}) \boldsymbol{\xi}^t \tag{16}$$

respectively; the matrix in the second parenthesis keeps the weight vectors normalized to unit length, assuming a small value for the learning rate parameter $\delta^t$ [8]. The dot-product metric form of a unit activity is

$$\eta_i^t = \frac{\exp\left[\beta \left(\sum_{j=1}^K \phi_j^i \boldsymbol{\mu}_j\right)^T \boldsymbol{\xi}^t\right]}{\sum_{i'=1}^K \exp\left[\beta \left(\sum_{j=1}^K \phi_j^{i'} \boldsymbol{\mu}_j\right)^T \boldsymbol{\xi}^t\right]} \tag{17}$$

which approximates the posterior probability $p(\boldsymbol{\zeta}_i|\boldsymbol{\xi}^t; \mathbf{M}, \beta)$ that the data vector were generated by that specific unit. This is based on the observation that if the data vectors $\{\boldsymbol{\xi}^t\}$ are normalized to unit length, the density generated in the data space (unit sphere in $\mathbf{R}^D$) becomes

$$p(\boldsymbol{\xi}|\boldsymbol{\zeta}_i; \mathbf{M}, \beta) = \left(\begin{array}{c}\text{normalizing} \\ \text{constant}\end{array}\right)^{-1} \times \exp\left[\beta \left(\sum_{j=1}^K \phi_j^i \boldsymbol{\mu}_j\right)^T \boldsymbol{\xi}\right] \tag{18}$$

### 3.2 S-Map algorithm minimizes the GTM error function in dot-product metric

The GTM error function is the negative log likelihood, which is given by (6) and is reproduced here:

$$\mathcal{E}(\mathbf{M}, \beta) = -\sum_{t=1}^T \ln\left[\frac{1}{K} \sum_{i=1}^K p(\boldsymbol{\xi}^t|\boldsymbol{\zeta}_i; \mathbf{M}, \beta)\right] \tag{19}$$

When the weights are updated using a batch version of (15), accumulating the updates for one epoch, the expected value of the error [4] for the unit $\boldsymbol{\zeta}_i$ is

$$\mathrm{E}(\mathcal{E}_i^{\text{new}}) = -\sum_{t=1}^T \underbrace{p^{\text{old}}(\boldsymbol{\zeta}_i|\boldsymbol{\xi}^t; \mathbf{M}, \beta)}_{\eta_i^{\text{old},t}} \ln[\underbrace{p^{\text{new}}(\boldsymbol{\zeta}_i)}_{=1/K} p^{\text{new}}(\boldsymbol{\xi}^t|\boldsymbol{\zeta}_i; \mathbf{M}, \beta)]$$

$$= -\sum_{t=1}^T \eta_i^{\text{old},t} \beta \left(\sum_{j=1}^K \phi_j^i \boldsymbol{\mu}_j^{\text{new}}\right)^T \boldsymbol{\xi}^t + \text{terms not involving the weight vectors} \tag{20}$$

The change of the error for the whole map after one epoch is thus

$$\mathrm{E}(\mathcal{E}^{\text{new}} - \mathcal{E}^{\text{old}}) = -\sum_{i=1}^K \sum_{t=1}^T \sum_{j=1}^K \eta_i^{\text{old},t} \beta \phi_j^i (\boldsymbol{\mu}_j^{\text{new}} - \boldsymbol{\mu}_j^{\text{old}})^T \boldsymbol{\xi}^t$$

$$= -\beta\delta \sum_{j=1}^K \underbrace{\left(\sum_{t=1}^T \sum_{i=1}^K \eta_i^{\text{old},t} \phi_j^i \boldsymbol{\xi}^t\right)^T}_{\boldsymbol{\sigma}_j^T} (\mathbf{I} - \boldsymbol{\mu}_j^{\text{old}} \boldsymbol{\mu}_j^{\text{old}\,T}) \underbrace{\left(\sum_{t'=1}^T \sum_{i'=1}^K \eta_{i'}^{\text{old},t'} \phi_j^{i'} \boldsymbol{\xi}^{t'}\right)}_{\boldsymbol{\sigma}_j} \tag{21}$$

$$= -\beta\delta \sum_{j=1}^K [\boldsymbol{\sigma}_j^T \boldsymbol{\sigma}_j - (\boldsymbol{\sigma}_j^T \boldsymbol{\mu}_j^{\text{old}})^2] \leq 0$$

with equality only when the weights are already in the error minimum.

## 4 Experimental results

The self-organization ability of the SOM, the GTM, and the S-Map was tested on an artificial data set: 500 points from a uniform random distribution in the unit square.

The initial weight vectors for all models were set to random values, and the final configuration of the map was plotted on top of the data (figure 1). For all the algorithms, the batch version was used. The SOM was trained as recommended in [1] – in two phases, the first starting with a wide neighborhood function, the second with a narrow neighborhood. The GTM was trained using the Matlab implementation by Svensén, following the recommendations given in [10]. The S-Map was trained in two ways: using the "full" rule (13), and the simplified rule (14). In both cases, the parameter $\beta$ value was slowly increased every epoch; by monitoring the error (6) of the S-Map (see the error plot in the figure) the suitable value for $\beta$ can be found.

In the GTM simulations, we experimented with many different choices for basis function width and their number, both with normalized and unnormalized basis functions. It turned out that GTM is somewhat sensitive to these choices: it had difficulties to unfold after a random initialization, unless the basis functions were set so wide (with respect to the weight matrix prior) that the map was well-organized already in its initial configuration. On the other hand, using very wide basis functions with the GTM resulted in a map that was too rigid to adapt well to the data. We also tried to update the parameter $\beta$ according to an annealing schedule, as with the S-Map, but this did not seem to solve the problem.

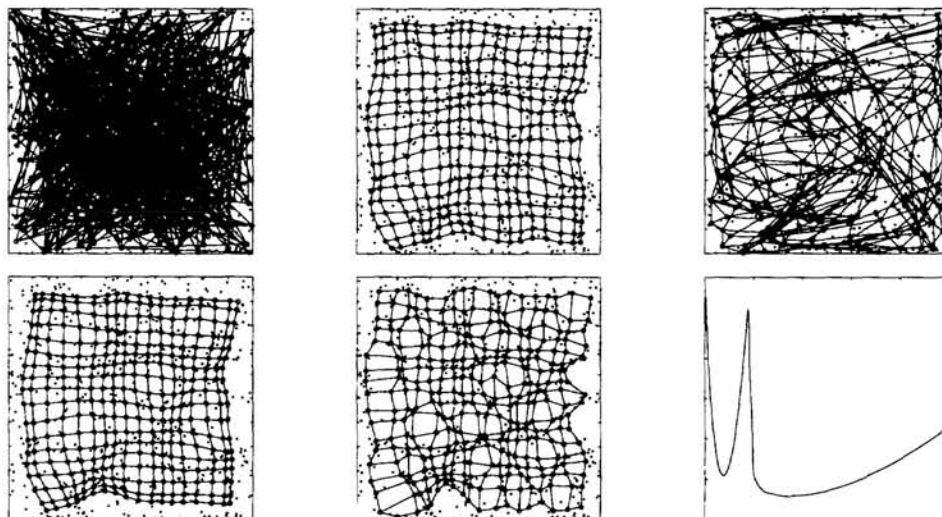

Figure 1: Random initialization (top left), SOM (top middle), GTM (top right), "full" S-Map (bottom left), simplified S-Map (bottom middle). On bottom right, the S-Map error as a function of epochs is displayed; the parameter $\beta$ was slightly increased every epoch, which causes the error to increase in the early (unfolding) phase of the learning, as the weight update only minimizes the error for a given $\beta$.

## 5 Conclusions

The S-Map and SOM seem to have a stronger tendency to self-organize from random initialization than the GTM. In data analysis applications, when the GTM can

be properly initialized, SOM, S-Map, and GTM yield comparable results; those obtained using the latter two algorithms are also straightforward to interpret in probabilistic terms. In Euclidean metric, the GTM has the additional advantage of guaranteed convergence to some error minimum; the convergence of the S-Map in Euclidean metric is still an open question. On the other hand, the batch GTM is computationally clearly heavier per epoch than the S-Map, while the S-Map is somewhat heavier than the SOM.

The SOM has an impressive record of proven applications in a variety of different tasks, and much more experimenting is needed for any alternative method to reach the same level of practicality. SOM is also the basic bottom-up procedure of self-organization in the sense that it starts from a minimum of functional principles realizable in parallel neural networks. This makes it hard to analyze, however. A probabilistic approach like the GTM stems from the opposite point of view by emphasizing the statistical model, but as a trade-off, the resulting algorithm may not share all the desirable properties of the SOM. Our new approach, the S-map, seems to have succeeded in inheriting the strong self-organization capability of the SOM, while offering a sound probabilistic interpretation like the GTM.

## Footnotes

[1]Note that this choice serves the purpose of illustration only; to use GTM properly, one should choose much more latent vectors than basis functions.

# References

[1] T. Kohonen, *Self-Organizing Maps.* Springer Series in Information Sciences 30, Berlin Heidelberg New York: Springer, 1995.

[2] T. Kohonen, E. Oja, O. Simula, A. Visa, and J. Kangas, "Engineering applications of the self-organizing map," *Proceedings of the IEEE*, vol. 84, pp. 1358–1384, Oct. 1996.

[3] C. M. Bishop, M. Svensen, and C. K. I. Williams, "GTM: A principled alternative to the self-organizing map," in *Advances in Neural Information Processing Systems* (to appear) (M. C. Mozer, M. I. Jordan, and T. Petche, eds.), vol. 9, MIT Press, 1997.

[4] A. P. Dempster, N. M. Laird, and D. B. Rubin, "Maximum likelihood from incomplete data via the EM algorithm," *Journal of the Royal Statistical Society*, vol. B 39, no. 1, pp. 1–38, 1977.

[5] S. P. Luttrell, "Code vector density in topographic mappings," Memorandum 4669, Defense Research Agency, Malvern, UK, 1992.

[6] T. M. Heskes and B. Kappen, "Error potentials for self-organization," in *Proceedings of the International Conference on Neural Networks (ICNN'93)*, vol. 3, (Piscataway, New Jersey, USA), pp. 1219–1223, IEEE Neural Networks Council, Apr. 1993.

[7] S. P. Luttrell, "A Bayesian analysis of self-organising maps," *Neural Computation*, vol. 6, pp. 767–794, 1994.

[8] E. Oja, "A simplified neuron model as a principal component analyzer," *Journal of Mathematical Biology*, vol. 15, pp. 267–273, 1982.

[9] K. Kiviluoto, "Topology preservation in self-organizing maps," in *Proceedings of the International Conference on Neural Networks (ICNN'96)*, vol. 1, (Piscataway, New Jersey, USA), pp. 294–299, IEEE Neural Networks Council, June 1996.

[10] M. Svensén, *The GTM toolbox – user's guide.* Neural Computing Research Group / Aston University, Birmingham, UK, 1.0 ed., Oct. 1996. Available at URL `http://neural-server.aston.ac.uk/GTM/MATLAB_Impl.html`.
